# Full-Distance Evasion of Pedestrian Detectors in the Physical World

**Zhi Cheng**[1], **Zhanhao Hu**[2], **Yuqiu Liu**[3], **Jianmin Li**[1], **Hang Su**[1], **Xiaolin Hu**[1*]

[1]Department of Computer Science and Technology, Tsinghua University, Beijing, China
[2]Department of Electrical Engineering and Computer Sciences, UC Berkeley
[3]Department of Technology, Beijing Forestry University, Beijing, China
zhicheng_mail@126.com, huzhanhao@berkeley.edu, yuqiu99@bjfu.edu.cn, {lijianmin, suhangss, xlhu}@mail.tsinghua.edu.cn

## Abstract

Many studies have proposed attack methods to generate adversarial patterns for evading pedestrian detection, alarming the computer vision community about the need for more attention to the robustness of detectors. However, adversarial patterns optimized by these methods commonly have limited performance at medium to long distances in the physical world. To overcome this limitation, we identify two main challenges. First, in existing methods, there is commonly an appearance gap between simulated distant adversarial patterns and their physical world counterparts, leading to incorrect optimization. Second, there exists a conflict between adversarial losses at different distances, which causes difficulties in optimization. To overcome these challenges, we introduce a Full Distance Attack (FDA) method. Our physical world experiments demonstrate the effectiveness of our FDA patterns across various detection models like YOLOv5, Deformable-DETR, and Mask RCNN. Codes available at https://github.com/zhicheng2T0/Full-Distance-Attack.git

## 1 Introduction

Currently, various adversarial attack methods have been proposed to evade deep-neural-network-based pedestrian detectors in the physical world [34, 17, 42] by crafting patches or clothes covered with adversarial patterns. These works have alarmed the computer vision community on the robustness of the existing Deep Nerual Network based detectors [37, 26, 14, 8, 43]. However, as shown in previous works [17, 23], a common limitation of the existing attack methods is that the generated adversarial patterns are not adversarially effective at medium to long distances (see also Figure 1(a)). This limitation might brings a false impression to the computer vision community that existing pedestrian detectors are robust to physical world attacks at such distances.

In this study, we find that the major cause of the aforementioned limitation is the naive distant image simulation technique used when optimizing the adversarial patterns. More specifically, as demonstrated in Figure 1(b), to simulate the appearance of a distant adversarial pattern during optimization, the existing attack algorithms usually naively downscale and apply the adversarial patterns according to the size of the pedestrians [34, 42, 17]. Such a naive technique creates a widening appearance gap between the simulated patterns and their real-world counterparts as distance increases. This leads to the optimization of incorrect adversarial patterns.

---

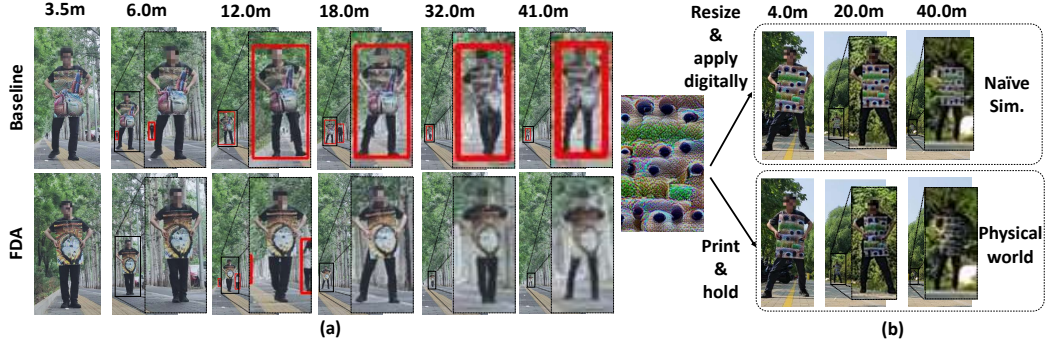

Figure 1: Illustrating FDA. (a) Visualizing attack performance of baseline [42] and FDA pattern. Red boxes are detection results from YOLOv5 [21] with confidence greater than 0.5. (b) Appearance gap between the naively simulated patch and its physical world counterparts at different distances.

To solve this problem, we propose a Distant Image Converter (DIC) to convert images of short-distance objects into an appearance similar to their physical world counterparts at long distances. In DIC, We find it necessary to simulate three factors in the physical world that contribute to the appearance gap. These factors include the effect of ***atmospheric perspective*** which changes object colors due to increasing scattering of light as distance increases, the effect of ***camera hardware*** which blurs the field of light projected from the target object to form a digital image, and the effect of the default ***effect filters*** commonly installed in digital cameras which change the color and texture details of the captured images for better visual appearances.

By applying the DIC during optimization, we found that different low frequency patterns were required at short and long distances, causing a conflict, hindering full distance attack (FDA) pattern optimization. To overcome the difficulty, we propose a Multi-Frequency Optimization (MFO) technique.

By combining DIC and MFO, we form the FDA method which generates effective adversarial patterns for evading pedestrian detectors at varying distances. Our physical world experiments demonstrate the effectiveness of our FDA patterns across various detection models like YOLOv5 [21], Deformable-DETR [43], and Mask RCNN [14].

## 2  Background and Related Work

**Atmospheric Perspective.** Atmospheric perspective refers to the phenomenon that as distance increases, the observed color of a target object exponentially shifts toward the color of the skylight (color of the sky in the direction of the object) due to the scattering of light by air molecules, dust and moisture as distance increases. Figure 2 (a) illustrates the phenomenon and Figure 2 (b) gives an intuitive example, where trees with a color of green and yellow appear blue at long distances due to atmospheric perspective.

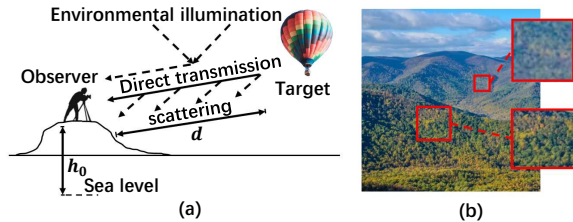

Figure 2: Atmospheric perspective. (a) Illustrating the phenomenon. (b) An example.

**Camera Imaging Pipeline.** To form an image, a camera receives an input light field and processes it through various lenses, including an anti-aliasing filter that blurs the light to prevent aliasing. Aliasing occurs when the camera's sensors naively sample the analog light field, leading to the incorrect recording of non-existent moiré patterns (e.g., Figure 3 (b)) [32]. The intuition of the phenomenon is illustrated in Figure 3 (c) with a 1-D example. That is, if the high-frequency blue curve is sampled at a low frequency, it can be inaccurately recorded as the red dotted curve with a wrong frequency. If a camera has a limited sampling rate, it may inaccurately sample high-frequency light information, resulting in moiré patterns. To prevent aliasing, anti-aliasing filters, or blurring filters, are commonly

applied before the imaging chip to filter out the high-frequency information that the imaging chip cannot correctly sample. In Figure 3 (d), an example image obtained by applying the anti-aliasing filter before sampling is demonstrated. After different lenses, the light field would pass through the aperture and shutter, and get sampled by the imaging chip with an array of rectangular light sensors. In the imaging chip, each sensor produces an RGB value by averaging the light projected onto it, resulting in an output image [32] that is further blurred relative to the field of light from the anti-aliasing filter.

**Effect Filters.** After obtaining a digital image with the imaging chip, digital cameras typically apply a variety of effect filters to enhance the visual appeal of the images [32]. Common effect filters include brightness, saturation, sharpening, exposure, contrast, highlight, shadow, vibrance, color temperature and so on. See Figure 3 (d) and (e) for visualization on the effect of applying the sharpening and contrast filters.

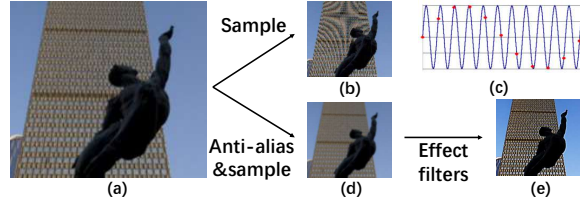

Figure 3: Factors influencing image effect. (a) Original light field, represented with an image. (b) Aliasing effects (simulated by applying sampling). (c) 1D aliasing. (d) Applying anti-aliasing before sampling. (e) Applying effect filters of sharpening and contrast. Images are from [32].

**Physical World Attacks with Adversarial Pattern.** A well-known limitation to deep learning models [13, 6, 30, 37, 26, 14, 8, 43] is that they are vulnerable to adversarial attacks [39, 9, 5, 28, 2]. Such a limitation makes crafting adversarial patterns effective for evading pedestrian detectors possible. Adv-Patch [34] is one of the earliest works that discovered adversarial patterns can disrupt detector decisions in the physical world. After that, many methods have been proposed to keep adversarial patterns effective when printed onto clothing (Adv-Tshirt [42]), to improve adversarial clothing performance at different angles (TCA [17]) and to improve naturalness of the adversarial clothing [33, 16, 18, 10]. In addition, many physical adversarial attack methods have been proposed for attacking vehicle detectors [35, 20, 7, 31, 15, 40], person re-identification models [38] and object tracking models[41, 4]. However, to the best of our knowledge, few works have addressed the decline of attack performance when distance increases.

## 3   Distant Image Converter

To bridge the appearance gap between distant adversarial pedestrian images in the physical and digital worlds, we propose to implement a Distant Image Converter (DIC). An intuitive solution is to train multiple neural networks, each specialized for image conversion at specific distances. However, this approach demands a large training set due to the large amount of parameters involved. In this work, we address this by implementing a physics-based DIC, leveraging principles of physics and camera hardware design. This ensures realistic image conversion with only 15 learnable parame-

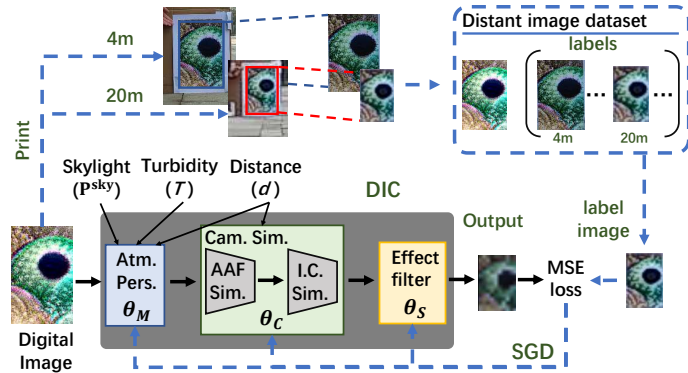

Figure 4: The overall pipeline of the DIC. The continuous arrows indicate the inference pipeline. The dashed arrows are used during training.

ters. Specifically, when given an input image, target distance, and environmental parameters like skylight RGB and turbidity values, the DIC should produces an output image that simulates the visual effect of positioning the input image at the target distance.

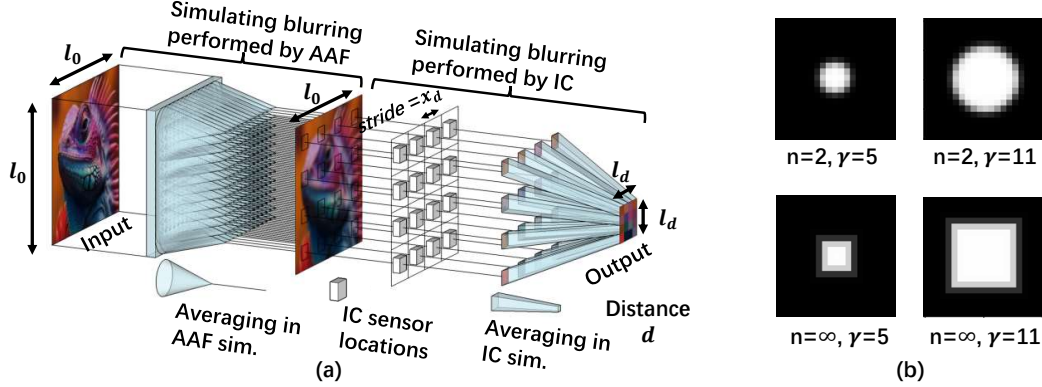

Figure 5: Camera simulation module. (a) An illustration of the blurring operations performed by the camera simulation module. (b) Visualizing outputs of function $\tilde{f}$ with different parameters.

The inference pipeline of our DIC is illustrated in Figure 4. That is, the DIC converts an input image with the *atmospheric perspective* module, the *camera simulation* module, and the *effect filter* module, simulating the physical world factors contributing to the appearance gap in sequence. We make the unknown physical world parameters controlling the effect of each module $\boldsymbol{\theta}_A, \boldsymbol{\theta}_C, \boldsymbol{\theta}_E$ learnable so that they can be effectively estimated through Stochastic Gradient Descent (SGD) with some training data. In the following, we first introduce each module then the training method.

## 3.1 Atmospheric Perspective Module

As demonstrated in Figure 2 (a), when an observer takes photos of a distant target at distance $d$, the ray of lights emitted from the target toward the observer (represented by image $\mathbf{I}^{\mathrm{obj}} \in \mathbb{R}^{3 \times w \times h}$, $w$ and $h$ being image width and height respectively) would be scattered away by molecules in the air, causing a decrease in brightness. At the same time, skylight with RGB values of $\mathbf{P}^{\mathrm{sky}} \in \mathbb{R}^3$ would be scattered toward the observer, shifting the target object's observed color towards the skylight.

To simulate the effect, we implement an atmospheric perspective module $F^{\mathrm{A}}$ following previous works [29, 12]. The inputs of the module include image $\mathbf{I}^{\mathrm{obj}}$ of the target object, a turbidity value $T$ representing air quality, the target distance $d$ to convert the image, and the skylight RGB value $\mathbf{P}^{\mathrm{sky}}$. The module produces an output image $\mathbf{I}_d^{\mathrm{M}}$ that simulates the effect of the target object at distance $d$ by processing $\mathbf{I}^{\mathrm{obj}}$ with

$$\mathbf{I}_d^{\mathrm{A}} = F^{\mathrm{A}}(\mathbf{I}^{\mathrm{obj}}, T, d, \mathbf{P}^{\mathrm{sky}}, \boldsymbol{\theta}_A) = \mathbf{I}^{\mathrm{obj}} e^{-\beta(\boldsymbol{\theta}_A, T)d} + tile(\mathbf{P}^{\mathrm{sky}})(1 - e^{-\beta(\boldsymbol{\theta}_A, T)d}), \tag{1}$$

where $tile$ is the tiling function that repeats $\mathbf{P}^{\mathrm{sky}}$ to form an image with the same shape of $\mathbf{I}^{\mathrm{obj}}$, $\boldsymbol{\theta}_A$ represents the estimated physics-related parameters and the $\beta$ function represents the decay rate. See Appendix A for more details on $\boldsymbol{\theta}_A$ and $\beta$.

## 3.2 Camera Simulation Module

To simulate the blurring effect introduced by the Anti-Aliasing Filter (AAF) and the Imaging Chip (IC) in camera, we model the camera using two convolutional layers in sequence. The blurring done by the layers is illustrated in Figure 5(a).

**Convolutional Layer Kernel Generation.** The AAF blurs each ray of light with neighboring light rays within a certain radius and the sensors on the IC to form their outputs by averaging all light rays projected onto them. To simulate their effects, convolutional layers with blurring kernels that take averages within circular and square regions in their local input windows at each stride should be used [32]. Additionally, since the camera simulation module needs to approximate the unknown AAF blurring strength and IC sensor size of different target cameras, the amount of pixels averaged by the kernels should be learnable through back-propagation.

To achieve this goal, although there are many alternative ways, we found that an effective approach is to implement a unified kernel generation function $f$ for both the AAF and IC simulation layers.

That is, as illustrated in Figure 5(b), we first implement a function $\tilde{f}$, which outputs values close to 1 when the $L^n$ norm of the coordinate $(i, j)$ within the kernel is less than $\frac{\gamma}{2}$ and outputs values near 0 otherwise:

$$\tilde{f}(\gamma, i, j, n) = \text{sigmoid}((i^n + j^n)^{\frac{1}{n}} + \frac{\gamma}{2}) * 3 + \text{sigmoid}(\frac{\gamma}{2} - (i^n + j^n)^{\frac{1}{n}}) * 3 - 1. \quad (2)$$

Then, we take the normalized version of $\tilde{f}$ as the function $f$ that generates the $j^{th}$ value in the $i^{th}$ row of the kernel weight $\mathbf{w} \in \mathbb{R}^{k \times k}$. That is, we let

$$\mathbf{w}_{i,j} = f(\gamma, k, i, j, n) = \frac{\tilde{f}(\gamma_d, i, j, n)}{\sum_{i=-k}^{k} \sum_{j=-k}^{k} \tilde{f}(\gamma_d, i, j, n)}. \quad (3)$$

By setting $n = 2$, the generated kernel takes averages within circular regions in its local input windows at each stride, allowing it to approximate the blurring done by the AAF. By setting $n = \infty$, the kernel takes averages within rectangular regions in its local input windows at each stride, mimicking the blurring done by the IC sensors. Moreover, by adjusting the learnable parameter $\gamma$, the number of pixels averaged by the kernel within its local input windows at different strides is altered, allowing for the simulation of various blurring strengths in the AAF and sensor sizes in the IC.

**Simulating AAF.** At a target distance $d$, to simulate the effect of the AAF averaging every incoming light ray with neighboring light rays within its blurring radius, we set the stride of the corresponding convolutional layer to 1, and generate the different entries of the kernel $\mathbf{w}_d^A$ within the layer using the function $f$ with $n = 2$. That is, we let

$$\mathbf{w}_{d,i,j}^A = f(\gamma_d^A, k_d^A, i, j, 2). \quad (4)$$

**Simulating Imaging Chip.** Similarly, we simulate the effect of IC at target distance $d$ with another convolutional layer. To simulate the effect that when a digital image with height $l_0$ is placed at a long distance $d$, the imaging chip would capture it with fewer sensors to form a smaller image with height $l_d$, as illustrated in Figure 5(a), we set the stride of the convolutional layer to $x_d = l_0/l_d$. To simulate the blurring done by the square imaging sensors, we generate different entries of its kernel $w_d^C$ using function $f$ with $n = \infty$, that is

$$\mathbf{w}_{d,i,j}^C = f(\gamma_d^C, k_d^C, i, j, \infty). \quad (5)$$

**Implementation of camera blurring strengths $\gamma_d^A$ and $\gamma_d^C$.** To set proper blurring strength $\gamma_d^A$ and $\gamma_d^C$ for simulating the AAF and IC at different distances, we observe that in the physical world, more distant objects are smaller relative to the fixed physical blurring strength in AAF ($\tilde{\gamma}^A$) and IC ($\tilde{\gamma}^C$). That is, the blurring strength ($\tilde{\gamma}$) to image height ($l_d$) ratio increases as distance increases.

In the digital world, when given a digital input image with height $l_0$, to consistently simulate the aforementioned ratio at different distances, we set $\gamma_d^A = \tilde{\gamma}^A \times x_d$ and $\gamma_d^C = \tilde{\gamma}^C \times x_d$. In this way, the blurring strength to image height ratio in the digital world ($\gamma_d/l_0 = \tilde{\gamma} \times x_d/l_0$) is identical to that in the physical world ($\tilde{\gamma}/l_d$) since $l_d = l_0/x_d$. A more detailed explanation with visualization is provided in Appendix B.

By setting $\boldsymbol{\theta}_C = [\tilde{\gamma}^A, \tilde{\gamma}^C]$ and optimizing $\boldsymbol{\theta}_C$, the camera simulation module can be trained to approximate the blurring effect of any target camera.

**Implementation of camera simulation module.** We express the camera simulation module $F^C$ with two channel-wise convolutional layers $\phi$ in sequence simulating the effect of AAF and IC at distance $d$ as

$$\mathbf{I}_d^C = F^C(\mathbf{I}_d^A, d, \mathbf{w}_d^A, \mathbf{w}_d^C) = \phi(\phi(\mathbf{I}_d^A, s = 1, w = \mathbf{w}_d^A), s = x_d, w = \mathbf{w}_d^C), \quad (6)$$

where $\mathbf{I}_d^A$ is the output of the atmospheric perspective module, $s$ and $w$ indicate the stride and kernel size of the two convolutional layers.

### 3.3 Effect Filter Simulation Module

In this module, we let $f_e$ denote the mapping function of a specific effect filter, and $F^E$ denote the mapping performed by a sequence of effect filters. We define $F^E$ as:

$$\mathbf{I}_d^E = F^E(\mathbf{I}_d^C, \boldsymbol{\theta}_E) = f_e(...f_1(\mathbf{I}_d^C, \theta_1), ..., \theta_e), \quad (7)$$

where $\mathbf{I}_d^C$ is the output of the camera simulation module, $\boldsymbol{\theta}_E = [\theta_1, ..., \theta_e]$ is the learnable parameter that controls the output effect of each effect filter, $\mathbf{I}_d^C$ is the input RGB image from the camera simulation module and $\mathbf{I}_d^E$ is the output image. By identifying the effect filters commonly involved in the target cameras and obtaining appropriate value for $\boldsymbol{\theta}_E$, the influence of effect filters can be simulated. Please refer to Appendix C for the exact computation of each effect filter.

### 3.4 Distant Image Converter Training

We collect a small distant image dataset and optimize the DIC to obtain appropriate DIC parameter values $(\boldsymbol{\theta}_A, \boldsymbol{\theta}_C, \boldsymbol{\theta}_E)$. As illustrated in Figure 4, we first printed images on papers, then by photographing the printed images at different distances and extracting crops that capture the same content as the original digital world images, we collected pairs between digital world images and their distant versions in the physical world. With this dataset, the parameters of the DIC can be optimized with stochastic gradient descent (SGD) using Mean Square Error (MSE) loss as the objective function, where the MSE loss is calculated between the DIC outputs and the corresponding physical world ground truth images with identical shape.

## 4 Full-Distance Attack

The optimization pipeline of the FDA method is illustrated in Figure 6. The adversarial patterns are first randomly cropped (as explained in Section 4.1) and applied onto short-distance pedestrian images. With the sub-patches applied, the pixels of the adversarial pedestrian are extracted (e.g. based on pedestrian masks generated with segmentation models). The extracted pixels are transformed by the DIC into their distant counterpart and superimposed onto randomly selected background images, generating a batch of distant adversarial pedestrian images. Within the same batch, different pre-selected distances are used to perform the distant image conversion. The parameter $T$ and $\mathbf{I}^{\text{sky}}$

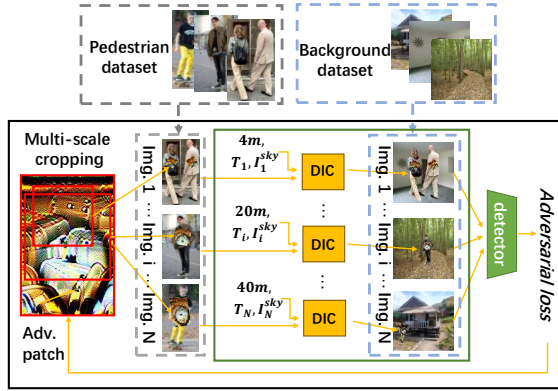

Figure 6: FDA overall optimization pipeline. $T_i$ and $\mathbf{I}_i^{\text{sky}}$ are randomly sampled turbidity and skylight values.

for DIC (from Equation (1)) are obtained by random sampling, allowing the FDA pattern optimized to be effective not only under different distances, backgrounds but also under different atmospheric conditions. Also, we apply EOTs [1] on $\boldsymbol{\theta}_A$, $\boldsymbol{\theta}_C$ and $\boldsymbol{\theta}_E$ to make the resulting FDA pattern robust against potential physical world disturbances. By feeding the current batch of adversarial pedestrian images into the target detector and calculating the loss function for performing the adversarial attack, the FDA pattern can be optimized through SGD.

At $D$ different simulated distances, we minimize the confidence $c$ and IOU $u$ of all $K$ correctly predicted pedestrian bounding boxes generated by the target detector. That is,

$$\mathcal{L}_{\text{adv}} = \sum_{d=1}^{D} \sum_{i=1}^{N_d} (\frac{\lambda_c}{K} \sum_{k=1}^{K} c_{i,k}^d + \frac{\lambda_u}{K} \sum_{k=1}^{K} u_{i,k}^d), \tag{8}$$

where $\lambda_c$ and $\lambda_u$ are manually set parameters balancing the two terms, $N_d$ is the number of images generated with adversarial pedestrians at distance $d$.

Please note that, without the green box part, the pipeline in Figure 6 degenerates to the Adv-Tshirt pipeline [42] with the only difference being that we sample random sub-patches of different sizes.

### 4.1 Multi-Frequency Optimization

Empirically, we found that by optimizing the FDA pattern with our proposed pipeline, there was a conflict in performance between short and long distances. We conjecture that this conflict might be due to the following two properties:

- At short distances, with a high resolution, both the low frequency patterns (or overall patterns) and high frequency patterns (or fine textures) can be optimized for performing attacks.

- At long distances, due to the reduced resolution, only the low frequency patterns remain, so only the low frequency patterns can be optimized for performing attacks.

A conflict could be resulted if the required low frequency patterns at short distances and long distances are different. To address this problem, we propose two Multi-Frequency Optimization (MFO) techniques to encourage the low frequency components of the adversarial patterns to be optimized for long distance attacks and the high frequency patterns to be optimized for short distance attacks.

**Multi-Scale Cropping (MSC) technique.** We set a smaller crop and patch application size for the adversarial patch when optimizing at short distances. In this way, it is harder for the short-distance optimization objectives to alter the overall pattern of the patch, preserving the low frequency patterns for long distance attacks.

**Two Stage Optimization (TSO) technique.** In TSO, we divide the optimization pipeline into two stages. In the first stage, the patch is optimized with a larger bias on long distances to obtain low frequency components for long distance attack by optimizing with more long distance images. In the second stage, the patch is optimized with a larger bias to obtain high frequency component for short distance attack by optimizing with more short distance pedestrian images. In addition, in the second stage, we add a loss function term that restrict the low-frequency components of the patch to remain unchanged. The loss function term introduced is described in Appendix D.

## 5  Experiments

**Subjects.** To evaluate the performance of different adversarial patterns in the physical world, we recruited five subjects (three males and two females with age ranging from 25 to 55) to collect test images and form demo videos. The recruitment and study procedures were approved by the Department of Psychology Ethics Committee, Tsinghua University, Beijing, China.

**Experiment settings.** Unless otherwise stated, we present our results with YOLOv5 [21] as the target model, the camera used to capture the physical world testing images was the back camera of Xiaomi-CIVI smart phone. For optimization details, we used configurations of Adv-Tshirt [42] and TCA [17] for patch and clothing experiments respectively. Given that all models failed to detect pedestrians reliably beyond around 41 meters, we optimized the adversarial patterns at simulated 4m, 8m, 14m, 20m, 26m, 34m and 40m, tested the patterns at 3.5m, 6m, 12m, 18m, 24m, 32m and 41m in the physical world unless otherwise stated. Appendix E provides a detailed analysis on the choice of the current distance range and the influence of holding a patch. To report reliable results, all physical world attack results reported are averaged over three trials, each trial had a different subject and a different location.

**Evaluation Metric.** We evaluated the performance of adversarial patterns with average attack success rate (ASR) across different distances. The ASR at a certain distance is defined as $1 - \frac{\text{TP}}{\text{GT}}$ where TP denotes the number of True Positives and GT denotes the number of Ground Truths. Following TCA [17], we set both the IOU and confidence thresholds for calculating TP to be 0.5.

**Distant Image Dataset.** To form a distant image dataset to train the DIC (Figure 4 (a)), we printed 45 training images and 9 testing images onto papers, collected photos of all printed images at 7 distances (4m, 8m, 14m, 20m, 26m, 34m, 40m) in 5 days and removed ones with noises (e.g. reflections and shadows). When photographing, skylight RGB values of the days were also recorded. Samples of training, test and skylight images are provided in Appendix F. We empirically found that though the dataset was small, it was enough to train a good DIC as the DIC only has 15 parameters (Section 5.1).

**Datasets for Pedestrian Attack.** To optimize the FDA patterns in the digital world, we created a pedestrian dataset and a background dataset. 1100 pedestrian images were extracted from existing datasets (INRIA [3], PennFudan [36] and COCO [24]). Additionally, we gathered 1000 more pedestrian images from online sources to boost diversity. We manually selected all images to ensure they had a similar resolution and scale to the 4-meter pedestrian images. Samples from the dataset are provided in Appendix F. We used the background dataset provided by an existing work [18]. To

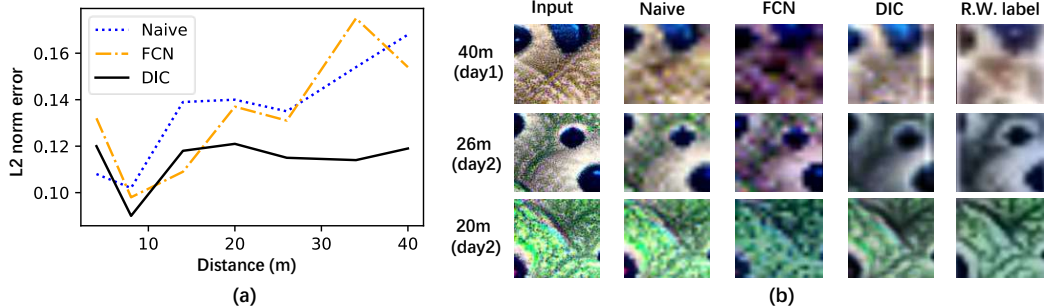

Figure 7: DIC results. (a) Performance of different distant image conversion methods on the test set of the distant image dataset. (b) Visualization of distant image conversion results. R.W. indicates real world.

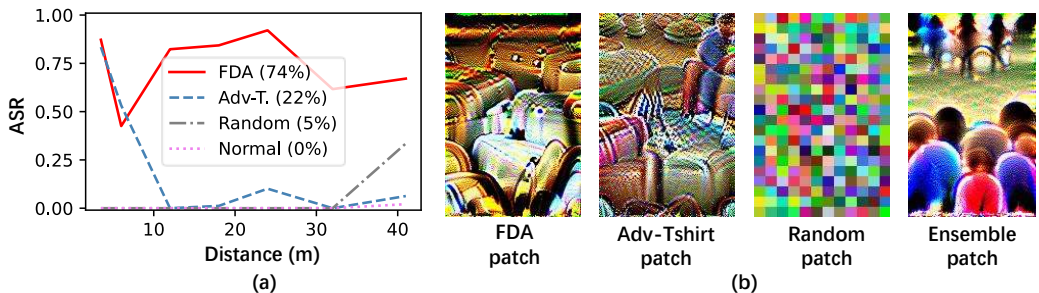

Figure 8: Physical world patch attack results. (a) ASR of different attack methods. Average ASRs of each curved reported in the brackets of the legend. (b) Adversarial patterns generated by different methods.

eliminate the potential for FDA patterns to over-fit to elements in our local environment, we excluded self-collected background images and self-collected 4-meter pedestrian images during optimization.

**Digital Patch Attack Evaluation Configuration.** To evaluate the average ASR of a patch in the digital world, we applied the patch onto 300 held out testing pedestrian images and converted them into their distant versions at different distances as in the training pipeline in Figure 6. By feeding the distant adversarial pedestrian images into the detector, the digital world ASRs of the patterns could be calculated.

## 5.1 DIC Results

We compared the performances of DIC and the naive distant image conversion method (achieved through image size reduction). In addition, we trained seven Fully Convolutional Network (FCN) [27] to convert distant images, each designed for a specific distance (See Appendix G for details). We used the $l_2$-norm error between labeled images from the physical world and the conversion results as the evaluation metric. As illustrated in Figure 7 (a), the DIC obtained a stable $l_2$-norm error of around 0.11 across different distances, where the $l_2$-norm error of the FCN and the naive method increased from around 0.11 to 0.16 as distance increased.

Figure 7 (b) visualizes the results of different methods. At different distances and on different days, it can be observed that compared to the conversion results generated by the naive method and the FCN method, the results generated by the DIC had the closest visual effect to the real world images.

## 5.2 Adversarial Patch Attack in the Physical World

**Settings.** Using adversarial patches, we evaluated the performance of patterns optimized with our FDA method against patterns obtained with different baseline methods in the physical world. For all patterns, we set a patch size of $200 \times 133$ and printed it onto a piece of paper with a size of 72 cm $\times$

| Source \ Target | YOLOv5 [21] | Mask RCNN (ResNet) [14] | Mask RCNN (SWIN)[26] | FrRCNN [8] | RetinaNet (PVT)[37] | YOLOv8 [22] | D-DETR [43] |
|---|---|---|---|---|---|---|---|
| Random | 5% | 7% | 12% | 9% | 19% | 4% | 6% |
| YOLOv5 | *74%* | 34% | 29% | 26% | 34% | 49% | **63%** |
| Ensemble | **79%** | *74%* | *71%* | **77%** | **87%** | **80%** | **75%** |

Table 1: Physical world black-box attack results. *Source* indicates the models that the FDA patterns were optimized against, *target* indicates the model attacked by the FDA patterns, *random* indicates random pattern and *ensemble* indicates ensemble attack. Black-box attack results with average ASRs $\geq 50\%$ are highlighted in bold face. White-box attack results are indicated with blue italic font.

50 cm. Each patch was tested at seven distances between 3.5 and 41 meters, with about 30 images collected at each distance within each trail (out of three trails) to calculate distance specific ASRs.

**Main Results.** Figure 8 (a) shows the ASR of our (YOLOv5) FDA pattern with respect to distance. For comparison, we also plot the results of the normal pedestrian (without holding an adversarial patch), a random pattern (formed by random RGB blocks) and an Adv-Tshirt pattern [42]. The patterns generated are visualized in Figure 8 (b). The FDA pattern achieved the highest average ASR of 74%. The corresponding digital world evaluation result is presented in Appendix H. We have also reproduced NAP[16] and T-SEA[19] and tested the patterns under comparable setting. The two patterns obtained average ASRs of 19% and 42% respectively. For visualization on FDA performance, we included demo videos in the supplementary material.

**Generalizing Across Scenarios.** When we used the back camera of Huawei-Nova-11-SE and OPPO-A9 smart phones when obtaining the testing images, the FDA pattern obtained average ASRs of 68% and 72% respectively. When the FDA pattern was tested under eight new test distances neighboring our original test distances, it obtained an average ASR of 76% (More details are provided in Appendix I). Such results demonstrate that the FDA pattern generalizes well across different scenarios.

**Adv-Tshirt with EOTs.** To investigate if it is possible to achieve FDA with a strong EOT [1] that might cover the effect of increased distance, we optimized three Adv-Tshirt patterns with EOTs in color and noise that were 1 time, 3 times and 10 times larger relative to the original strength. The corresponding patches only obtained average ASRs of 22%, 10% and 8% respectively, demonstrating that a larger EOT is not helpful for performing FDA.

**Ablation Studies.** To evaluate the influence of different design components, we first removed both DIC and MFO. The resulting FDA pattern obtained an average ASR of 22%. By adding the DIC and MFO back into the optimization pipeline, the average ASR of the resulting patterns increased to 65% and 74% respectively, indicating that they both contributed to the performance of FDA. Analysis on the influence of different components in DIC and MFO, together with analysis on the presence of conflict in attack performance between short and long distances are provided in Appendix J.

**Black-Box Attacks.** To investigate the generalizability of FDA patterns across detectors, we transfered the FDA pattern optimized for YOLOv5 to attack 6 black-box models (as shown in Table 1). Without specific designs for boosting transferability, the FDA pattern did not achieve good ASRs except on the Deformable DETR. We then integrated FDA with ensemble attack [25] (by optimizing the FDA pattern to be effective for both Mask RCNNs with ResNet backbone [14] and Swin backbone [26]), the resulting FDA pattern achieved good black-box attack performance, obtaining average ASRs of more than 75% on all black-box models. The FDA pattern optimized for ensemble attack is visualized in Figure 8 (b).

## 5.3 Clothing Attacks in the Physical World

To optimize the FDA pattern for clothing attacks, we incorporated toroidal cropping [17] into the FDA pipeline. We followed the process outlined in Figure 6, with the sole modification that we tiled the adversarial patch prior to multi-scale cropping by tripling the patch both vertically and horizontally. By integrating toroidal cropping, we enabled the FDA clothing to attack from all angles as for the TCA clothing (we leveraged TCA instead of TC-EGA since we found the TCA method to be more effective on YOLOV5). Different clothing tested targeting YOLOV5 are illustrated in Figure 9 (b). The clothing tailoring process and different adversarial patterns tested are provided in Appendix K.

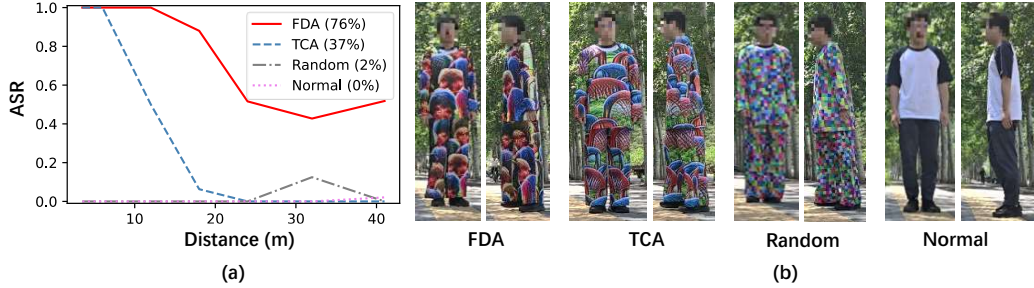

Figure 9: Clothing attack. (a) Attack results. (b) Front and side view of different clothing.

In the experiments, we tested the FDA clothing, TCA clothing [17], random clothing and normal clothing. Each clothing was tested at seven distances between 3.5 and 41.0 meters, with about 30 images collected within each trail (out of three trails) at each distance to calculate the ASRs.

As demonstrated in Figure 9 (a), when targeting the YOLOV5 [21] model, in the front and back view, both the random clothing and the normal clothing had near zero attack performance. By covering the entire body of the subject, the TCA clothing obtained an average ASR of 37%. The FDA clothing outperformed the TCA clothing with an average ASR of 76%. Similarly, in the side view, the FDA clothing obtained an average ASR of 61%, being 15% higher than TCA, while the random and normal clothing had average ASRs of 0%.

Additionally, when treating the Deformable DETR [43] and the RetinaNet with PVT backbone [37] as the target model, in the front and back view, the FDA clothing has also attacked the target models effectively by obtaining average ASRs of 71% and 73% respectively.

If we calculate the mean average ASRs across confidence thresholds of 0.1, 0.2, ..., 0.9, when targeting YOLOV5, in the front and back view, the mean average ASR of the FDA clothing, TCA clothing, random clothing and normal clothing was 78%, 43%, 9% and 3% respectively. Analysis on the performance of different FDA clothing under different IOU thresholds is included in Appendix K.

The results confirm that our proposed method has the ability to boost FDA performance at all angles and the FDA method is effective for clothing attacks.

## 6 Limitation and Potential Social Impact

As described in Appendix K, depending on the target model and attack method used, if abstract human-like patterns appear on the FDA pattern, the detector may generate small pedestrian bounding boxes for these patterns, even when the actual pedestrian subjects are not detected. This could lead to a decrease in FDA performance on some models when a smaller IOU threshold is applied. However, this is not specific to the FDA method as human-like patterns have appeared in previous attack methods [34, 17].

Physical-world adversarial attack research can lead to unwanted applications in real-world scenarios such as evading security cameras, but we publish our work to inspire researchers to propose more reliable detectors with adversarial defense mechanisms. We also urge readers and future researchers to set strict access controls for adversarial patterns and pattern generation codes targeting detectors in security-sensitive areas. This may include user authentication, licensing agreements, and usage monitoring.

## 7 Conclusion

In this work, we bridge the appearance gap between the digital world and the physical world by proposing the DIC. Moreover, we avoid conflicts that impede optimization by proposing two MFO optimization techniques. Together, the FDA patterns gained high ASRs targeting different detectors within a wide range of distances in the physical world.

## Acknowledgments

This work was supported by the National Natural Science Foundation of China under grants U2341228 and 92370124.

## References

[1] A. Athalye, L. Engstrom, A. Ilyas, and K. Kwok. Synthesizing robust adversarial examples. In J. G. Dy and A. Krause, editors, *Proceedings of the 35th International Conference on Machine Learning, ICML 2018, Stockholmsmässan, Stockholm, Sweden, July 10-15, 2018*, volume 80 of *Proceedings of Machine Learning Research*, pages 284–293. PMLR, 2018.

[2] N. Carlini and D. A. Wagner. Towards evaluating the robustness of neural networks. In *2017 IEEE Symposium on Security and Privacy, SP 2017, San Jose, CA, USA, May 22-26, 2017*, pages 39–57. IEEE Computer Society, 2017.

[3] Y.-T. Chen, C.-S. Chen, Y.-P. Hung, and K.-Y. Chang. Multi-class multi-instance boosting for part-based human detection. *2009 IEEE 12th International Conference on Computer Vision Workshops, ICCV Workshops*, pages 1177–1184, 2009.

[4] L. Ding, Y. Wang, K. Yuan, M. Jiang, P. Wang, H. Huang, and Z. J. Wang. Towards universal physical attacks on single object tracking. *Proceedings of the AAAI Conference on Artificial Intelligence*, 35(2): 1236–1245, May 2021.

[5] Y. Dong, F. Liao, T. Pang, X. Hu, and J. Zhu. Discovering adversarial examples with momentum. *CoRR*, abs/1710.06081, 2017. URL `http://arxiv.org/abs/1710.06081`.

[6] A. Dosovitskiy, L. Beyer, A. Kolesnikov, D. Weissenborn, X. Zhai, T. Unterthiner, M. Dehghani, M. Minderer, G. Heigold, S. Gelly, J. Uszkoreit, and N. Houlsby. An image is worth 16x16 words: Transformers for image recognition at scale. *CoRR*, abs/2010.11929, 2020. URL `https://arxiv.org/abs/2010.11929`.

[7] Y. Duan, J. Chen, X. Zhou, J. Zou, Z. He, J. Zhang, W. Zhang, and Z. Pan. Learning coated adversarial camouflages for object detectors. In L. D. Raedt, editor, *Proceedings of the Thirty-First International Joint Conference on Artificial Intelligence, IJCAI-22*, pages 891–897. International Joint Conferences on Artificial Intelligence Organization, 7 2022. Main Track.

[8] R. Girshick. Fast R-CNN. In *Proceedings of the IEEE International Conference on Computer Vision (ICCV)*, pages 1440–1448, December 2015.

[9] I. Goodfellow, J. Shlens, and C. Szegedy. Explaining and harnessing adversarial examples. In *International Conference on Learning Representations*, 2015. URL `http://arxiv.org/abs/1412.6572`.

[10] A. Guesmi, I. M. Bilasco, M. Shafique, and I. Alouani. Advart: Adversarial art for camouflaged object detection attacks. *ArXiv*, abs/2303.01734, 2023.

[11] D. M. Hanumantharaju, G. Vishalakshi, S. Halvi, and S. Satish. A novel fpga based reconfigurable architecture for image color space conversion. *Communications in Computer and Information Science*, 270, 12 2011.

[12] K. He, J. Sun, and X. Tang. Single image haze removal using dark channel prior. In *2009 IEEE Conference on Computer Vision and Pattern Recognition (CVPR)*, pages 1956–1963, 2009.

[13] K. He, X. Zhang, S. Ren, and J. Sun. Deep residual learning for image recognition. In *2016 IEEE Conference on Computer Vision and Pattern Recognition (CVPR)*, pages 770–778, 2016.

[14] K. He, G. Gkioxari, P. Dollar, and R. Girshick. Mask r-cnn. In *Proceedings of the IEEE International Conference on Computer Vision (ICCV)*, pages 2961–2969, Oct 2017.

[15] S. Hoory, T. Shapira, A. Shabtai, and Y. Elovici. Dynamic adversarial patch for evading object detection models. *ArXiv*, abs/2010.13070, 2020.

[16] Y.-C.-T. Hu, B.-H. Kung, D. S. Tan, J.-C. Chen, K.-L. Hua, and W.-H. Cheng. Naturalistic physical adversarial patch for object detectors. In *Proceedings of the IEEE/CVF International Conference on Computer Vision (ICCV)*, pages 7848–7857, October 2021.

[17] Z. Hu, S. Huang, X. Zhu, F. Sun, B. Zhang, and X. Hu. Adversarial texture for fooling person detectors in the physical world. In *Proceedings of the IEEE/CVF Conference on Computer Vision and Pattern Recognition (CVPR)*, pages 13307–13316, June 2022.

[18] Z. Hu, W. Chu, X. Zhu, H. Zhang, B. Zhang, and X. Hu. Physically realizable natural-looking clothing textures evade person detectors via 3d modeling. In *Proceedings of the IEEE/CVF Conference on Computer Vision and Pattern Recognition (CVPR)*, pages 16975–16984, June 2023.

[19] H. Huang, Z. Chen, H. Chen, Y. Wang, and K. Zhang. T-SEA: Transfer-based self-ensemble attack on object detection, 2022. URL `https://arxiv.org/abs/2211.09773`.

[20] L. Huang, C. Gao, Y. Zhou, C. Xie, A. L. Yuille, C. Zou, and N. Liu. Universal physical camouflage attacks on object detectors. In *IEEE/CVF Conference on Computer Vision and Pattern Recognition (CVPR)*, June 2020.

[21] G. Jocher. Ultralytics yolov5, 2020. URL `https://github.com/ultralytics/yolov5`.

[22] G. Jocher, A. Chaurasia, and J. Qiu. Ultralytics yolov8, 2023. URL `https://github.com/ultralytics/ultralytics`.

[23] T. Kim, Y. Yu, and Y. M. Ro. Multispectral invisible coating: Laminated visible-thermal physical attack against multispectral object detectors using transparent low-e films. *Proceedings of the AAAI Conference on Artificial Intelligence*, 37(1):1151–1159, Jun. 2023.

[24] T. Lin, M. Maire, S. J. Belongie, L. D. Bourdev, R. B. Girshick, J. Hays, P. Perona, D. Ramanan, P. Doll'a r, and C. L. Zitnick. Microsoft COCO: common objects in context. *CoRR*, abs/1405.0312, 2014. URL `http://arxiv.org/abs/1405.0312`.

[25] Y. Liu, X. Chen, C. Liu, and D. Song. Delving into transferable adversarial examples and black-box attacks. In *5th International Conference on Learning Representations, ICLR 2017, Toulon, France, April 24-26, 2017, Conference Track Proceedings*. OpenReview.net, 2017.

[26] Z. Liu, Y. Lin, Y. Cao, H. Hu, Y. Wei, Z. Zhang, S. Lin, and B. Guo. Swin transformer: Hierarchical vision transformer using shifted windows. In *Proceedings of the IEEE/CVF International Conference on Computer Vision (ICCV)*, pages 10012–10022, October 2021.

[27] J. Long, E. Shelhamer, and T. Darrell. Fully convolutional networks for semantic segmentation. In *The IEEE/CVF Conference on Computer Vision and Pattern Recognition (CVPR)*, June 2015.

[28] A. Madry, A. Makelov, L. Schmidt, D. Tsipras, and A. Vladu. Towards deep learning models resistant to adversarial attacks. In *6th International Conference on Learning Representations, ICLR 2018, Vancouver, BC, Canada, April 30 - May 3, 2018, Conference Track Proceedings*. OpenReview.net, 2018.

[29] C. Morales, T. Oishi, and K. Ikeuchi. Real-time rendering of aerial perspective effect based on turbidity estimation. *IPSJ Transactions on Computer Vision and Applications*, 9, 12 2017.

[30] J. Redmon, S. Divvala, R. Girshick, and A. Farhadi. You only look once: Unified, real-time object detection. In *In Proceedings of the IEEE/CVF Conference on Computer Vision and Pattern Recognition (CVPR)*, pages 779–788, Los Alamitos, CA, USA, jun 2016. IEEE Computer Society.

[31] N. Suryanto, Y. Kim, H. Kang, H. T. Larasati, Y. Yun, T.-T.-H. Le, H. Yang, S.-Y. Oh, and H. Kim. Dta: Physical camouflage attacks using differentiable transformation network. In *Proceedings of the IEEE/CVF Conference on Computer Vision and Pattern Recognition (CVPR)*, pages 15305–15314, June 2022.

[32] R. Szeliski. *Computer vision algorithms and applications*. Springer, London; New York, 2011. ISBN 9781848829343 1848829345 9781848829350 1848829353.

[33] J. Tan, N. Ji, H. Xie, and X. Xiang. Legitimate adversarial patches: Evading human eyes and detection models in the physical world. In *Proceedings of the 29th ACM International Conference on Multimedia*, MM '21, page 5307–5315, New York, NY, USA, 2021. Association for Computing Machinery. ISBN 9781450386517.

[34] S. Thys, W. V. Ranst, and T. Goedemé. Fooling automated surveillance cameras: Adversarial patches to attack person detection. In *CVPR Workshops*, pages 49–55. Computer Vision Foundation / IEEE, 2019.

[35] J. Wang, A. Liu, Z. Yin, S. Liu, S. Tang, and X. Liu. Dual attention suppression attack: Generate adversarial camouflage in physical world. In *Proceedings of the IEEE/CVF Conference on Computer Vision and Pattern Recognition (CVPR)*, pages 8565–8574, June 2021.

[36] L. Wang, J. Shi, G. Song, and I.-f. Shen. Object detection combining recognition and segmentation. In Y. Yagi, S. B. Kang, I. S. Kweon, and H. Zha, editors, *Computer Vision – ACCV 2007*, pages 189–199, Berlin, Heidelberg, 2007. Springer Berlin Heidelberg. ISBN 978-3-540-76386-4.

[37] W. Wang, E. Xie, X. Li, D.-P. Fan, K. Song, D. Liang, T. Lu, P. Luo, and L. Shao. Pyramid vision transformer: A versatile backbone for dense prediction without convolutions. In *Proceedings of the IEEE/CVF International Conference on Computer Vision (ICCV)*, pages 548–558, 2021.

[38] Z. Wang, S. Zheng, M. Song, Q. Wang, A. Rahimpour, and H. Qi. advpattern: Physical-world attacks on deep person re-identification via adversarially transformable patterns. In *Proceedings of the IEEE/CVF International Conference on Computer Vision (ICCV)*, October 2019.

[39] H. Wei, H. Tang, X. Jia, Z. Wang, H. Yu, Z. Li, S. Satoh, L. Van Gool, and Z. Wang. Physical adversarial attack meets computer vision: A decade survey. *IEEE Transactions on Pattern Analysis and Machine Intelligence*, pages 1–20, 2024.

[40] H. Wen, S. Chang, and L. Zhou. Light projection-based physical-world vanishing attack against car detection. In *ICASSP 2023 - 2023 IEEE International Conference on Acoustics, Speech and Signal Processing (ICASSP)*, pages 1–5, 2023.

[41] R. R. Wiyatno and A. Xu. Physical adversarial textures that fool visual object tracking. In *Proceedings of the IEEE/CVF International Conference on Computer Vision (ICCV)*, October 2019.

[42] K. Xu, G. Zhang, S. Liu, Q. Fan, M. Sun, H. Chen, P. Chen, Y. Wang, and X. Lin. Evading real-time person detectors by adversarial t-shirt. *CoRR*, abs/1910.11099, 2019. URL `http://arxiv.org/abs/1910.11099`.

[43] X. Zhu, W. Su, L. Lu, B. Li, X. Wang, and J. Dai. Deformable detr: Deformable transformers for end-to-end object detection. In *9th International Conference on Learning Representations, ICLR 2021, Virtual Event, Austria, May 3-7, 2021*, 2021.

## A    Details for the Atmospheric Perspective Simulation Module

Following existing formulations [29, 12], we calculate $\beta(\boldsymbol{\theta}_A, T)$ in Equation (1) by

$$\beta(\boldsymbol{\theta}_A, T) = \beta_R(\boldsymbol{\theta}_A, T) + \beta_M(\boldsymbol{\theta}_A, T), \tag{9}$$

where $\beta_R(\boldsymbol{\theta}_A, T)$ describes the scattering effect caused by air molecules and $\beta_M(\boldsymbol{\theta}_A, T)$ describes the scattering effect caused by moisture and dust. More specifically, $\beta_R(\boldsymbol{\theta}_A, T)$ and $\beta_M(\boldsymbol{\theta}_A, T)$ are:

$$\beta_R(\boldsymbol{\theta}_A, T) = \frac{8\pi^3(n^2 - 1)^2}{3N\lambda_i^4}\left(\frac{6 + 3p_n}{6 - 7p_n}\right)e^{-\frac{h}{H_{R0}}} \tag{10}$$

and

$$\beta_M(\boldsymbol{\theta}_A, T) = 0.434 \cdot (0.6544T - 0.6510) \cdot c \cdot \pi\left(\frac{2\pi}{\lambda_i}\right)^2 \cdot 0.67 \cdot e^{-\frac{h}{H_{M0}}}, \tag{11}$$

where $n = 1.0003$ is the refractive index of air in the visible spectrum, $N$ is the molecular density of air, $p_n = 0.035$ is the depolarization factor of air, $h$ is the altitude at the scattering point, $H_{R0} = 7994m$ is the scale height for Rayleigh scattering, $H_{M0} = 1200m$ is the scale height for Mie scattering, $\lambda_i$ is the wavelength captured to be the output of the current $i^{th}$ camera channel, $T$ is the turbidity and $c$ is an empirical parameter.

Since the molecular density $N$ depends on environmental conditions such as atmospheric pressure, we initialize $N$ with the molecular density of standard atmosphere $2.545 \times 10^{25} m^{-3}$ and set it to be learnable to estimate its value by optimization. Since imaging chips from different brands capture lights with different wavelengths as the RGB channel outputs, we set $\lambda_i$ representing the unknown light wavelength captured by camera channels to be a learnable parameter. Since the parameter $c$ has been estimated to be in vastly different values in different works [29], to estimate its value in our local physical world environment, we set it to be learnable. That is, we let $\boldsymbol{\theta}_A = [N, c, \lambda_R, \lambda_G, \lambda_B]$. In addition, we set the turbidity $T$ during DIC training and randomize $T$ during FDA optimization to improve the robustness of the FDA pattern toward turbidity changes in the physical world.

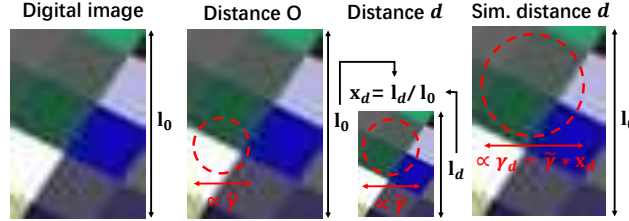

Figure S1: visualizing relationships between $\tilde{\gamma}$, $\gamma$, $x_d$, $l_0$ and $l_d$.

## B    More Details on $\tilde{\gamma}^A$ and $\tilde{\gamma}^C$

As illustrated in Figure S1, suppose we have a digital image with a height of $l_0$ pixels and when placed at distance $O$ in the physical world, the captured image also has a height of $l_0$ pixels. In the physical world, as distance increases to distance $d$, the height of the image would reduce to $l_d$. Since the AAF and IC in cameras are implemented by hardware, their physical blurring strength $\tilde{\gamma}^A$ and $\tilde{\gamma}^C$ stay identical regardless of $d$, so as the height of the image decreases to $l_d$ with increasing $d$, the ratio $\tilde{\gamma}/l_d$ increases.

In the digital world, when implementing the camera simulation module, our goal is to accurately simulate the camera's imaging pipeline of first performing blurring by AAF then down-sampling with IC. Thus, it is inappropriate to use a naive simulation pipeline such as first down-scaling the digital image to height $l_d$ and then applying AAF and IC simulation with blurring strengths $\tilde{\gamma}^A$ and $\tilde{\gamma}^C$, as it would introduce incorrect aliasing effects and sample incorrect information.

When the target distance is $d$, the digital image with height $l_0$ is $x_d$ times larger than the corresponding physical world image, to perform the simulation correctly, we apply blurring kernels with $x_d$ times larger blurring strength by letting $\gamma_d^A = \tilde{\gamma}^A \times x_d$ and $\gamma_d^C = \tilde{\gamma}^C \times x_d$ for the AAF and IC simulation layers. This ensures that the blurring strength to image scale ratio matches the corresponding scenario in the physical world, allowing the blurring effect to be correctly and consistently simulated across different distances.

## C    Details for Effect Filter Simulation Module

We used eight effect filters $f_e$ in the effect filter simulation module and optimized their parameters $\theta_e$. The parameters were initialized such that the filters approximate identity mappings.

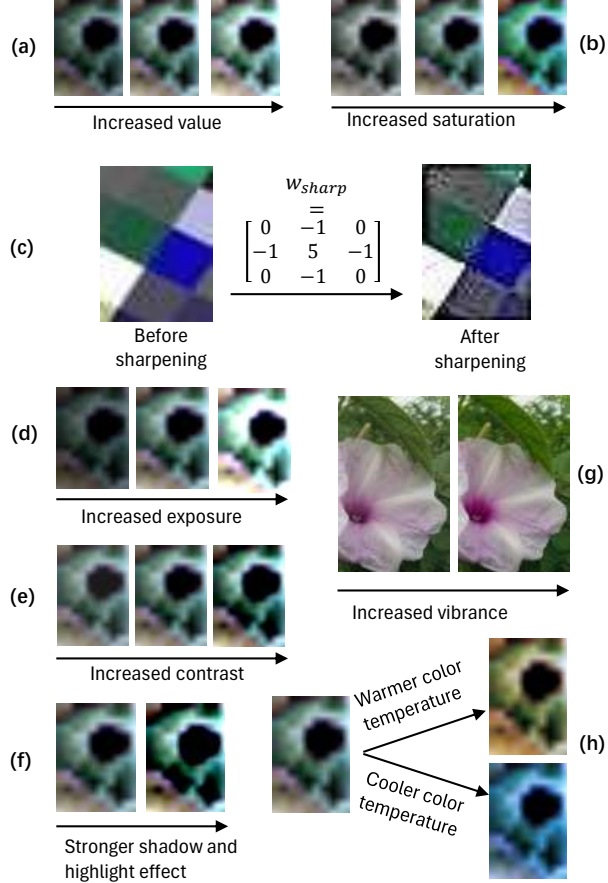

The first effect filter $f_1$ alters the brightness of images. It first converts the RGB image $\mathbf{I^{in}}$ into the HSV space (H for hue, S for saturation, V for value) [11], then multiplies the value of channel V by a learnable parameter $\theta_1$, and converts the resulting HSV values back to the RGB space. This filter performs the computation of

$$\mathbf{I^{out}} = f_1(\mathbf{I^{in}}, \theta_1)$$
$$= \text{RGB}(\text{Clamp}(\text{V}(\text{HSV}(\mathbf{I^{in}}), \theta_1))), \tag{12}$$

where $\mathbf{I^{out}}$ is the output image of the filter, $\text{HSV}(\cdot)$ is the function that converts RGB image into the HSV space [11], $\text{V}(\cdot)$ is the function that multiplies the value channel of the image in the HSV space by $\theta_1$, $\text{Clamp}$ is the function that clamps the HSV value to be within the range of [0,1] and $\text{RGB}(\cdot)$ is the function that converts image in HSV space back to the RGB space [11]. See Figure S2 (a) for visualization. With larger $\theta_1$, the output images are brighter.

The second effect filter $f_2$ is the saturation filter that alters the colorfulness of images. Similar to the value filter, the effect filter $f_2$ first converts the RGB image $\mathbf{I^{in}}$ into the HSV space, then multiplies the saturation channel by a learnable parameter $\theta_2$ and converts the resulting HSV values back to the RGB space. This filter performs the computation of

Figure S2: Effect of applying different effect filters.

$$\mathbf{I^{out}} = f_2(\mathbf{I^{in}}, \theta_2) = \text{RGB}(\text{Clamp}(\text{S}(\text{HSV}(\mathbf{I^{in}}), \theta_2))), \tag{13}$$

where $\text{S}(\cdot)$ is the function that multiplies the saturation channel of the image in the HSV space by $\theta_2$. See Figure S2 (b) for visualization. With larger $\theta_2$, the output images are more colorful.

The third effect filter $f_3$ is the sharpening filter. The filter transforms its input image $\mathbf{I^{in}}$ by performing two convolutions ($\phi$) in sequence, the first performing sharpening, the second performing blurring. The blurring layer has its blurring strength controlled by the learnable parameter $\theta_3$ so that the sharpness of the output image can be manipulated. More specifically, the computation is

$$\mathbf{I^{out}} = f_3(\mathbf{I^{in}}, \theta_3) = \phi(\phi(\mathbf{I_c}, s = 1, w = w_{sharp}), s = 1, w = w_{gaus}), \tag{14}$$

where $\phi(\cdot)$ is the channel-wise convolution operation, $s$ is the stride of the convolution operation, $w_{sharp}$ is the sharpening kernel demonstrated in Figure S2 (c), $w_{gaus}$ is the Gaussian kernel generated by function $g(\theta_3)$ and $\theta_3$ is the standard deviation of the Gaussian kernel. See Figure S2 (c) for visualization. With the sharpening filter applied, highlight and shadow bands are added onto the color boundaries of the input image.

The fourth effect filter $f_4$ is the exposure filter that increases the brightness of the input image $\mathbf{I^{in}}$ exponentially according to its learnable parameter $\theta_4$. This filter performs the computation of

$$\mathbf{I^{out}} = f_4(\mathbf{I^{in}}, \theta_4) = \mathrm{Clamp}(\mathbf{I^{in}} * 2^{\theta_4}). \tag{15}$$

See Figure S2 (d) for visualization. With larger $\theta_4$, more pixels would become exposed by having values closer to 1 in all channels.

The fifth effect filter $f_5$ is the contrast filter that increases the brightness gap between brighter and darker pixels according to the learnable parameter $\theta_5$. This filter performs the computation of

$$\mathbf{I^{out}} = f_5(\mathbf{I^{in}}, \theta_5) = \mathrm{Clamp}((\mathbf{I^{in}} - 0.502) * \frac{1.015 * (\theta_5 + 1)}{1.015 - \theta_5}). \tag{16}$$

See Figure S2 (e) for visualization. With $\theta_5$ increased, the brightness gap increases, which causes the details within the image to become more salient.

The sixth effect filter $f_6$ is the highlight and shadow filter. The effect of the filter is controlled by the learnable parameters $\theta_{6,\mathrm{low}}$ and $\theta_{6,\mathrm{high}}$. The filter sets all entries within the input image $\mathbf{I^{in}}$ that have values lower than $\theta_{6,\mathrm{low}}$ to be 0, all entries higher than $\theta_{6,\mathrm{high}}$ to be 1, and increases the contrast of entries that have values between $\theta_{6,\mathrm{low}}$ and $\theta_{6,\mathrm{high}}$. This filter performs the computation of

$$\mathbf{I^{out}} = f_6(\mathbf{I^{in}}, \theta_{6,\mathrm{low}}, \theta_{6,\mathrm{high}}) = \mathrm{Clamp}(\frac{\mathbf{I^{in}} - \theta_{6,\mathrm{low}}}{\theta_{6,\mathrm{high}} - \theta_{6,\mathrm{low}}}). \tag{17}$$

See Figure S2 (f) for visualization. With larger $\theta_{6,\mathrm{low}}$ and lower $\theta_{6,\mathrm{high}}$, it causes the darker and brighter details in the images to be more salient.

The seventh effect filter $f_7$ is the vibrance filter that alters the contrast in saturation. This filter forms the output by first transforming the input RGB image into the HSV space, then altering contrast in the S channel according to the learnable parameter $\theta_7$ and forming the output by converting the altered HSV values back to the RGB space. This filter performs the computation of

$$\mathbf{I^{out}} = f_7(\mathbf{I^{in}}, \theta_7) = \mathrm{RGB}(\mathrm{B}(\mathrm{HSV}(I^{in}), \theta_7)), \tag{18}$$

where function B is the function that processes each channel $\mathbf{I_i^{HSV}}$ of the input image in the HSV space independently ($i$ indicates the channel) and forms the corresponding output channels $\mathbf{\tilde{I}_i^{HSV}}$ following

$$\mathbf{\tilde{I}_i^{HSV}} = \mathrm{B}(\mathbf{I_i^{HSV}}) = \begin{cases} \mathbf{I_i^{HSV}}, \text{ if } i = H \text{ or } V \\ \frac{1}{(1+e^{-\theta_7 * (\mathbf{I_i^{HSV}} - 0.5))}}, \text{ if } i = S. \end{cases} \tag{19}$$

See Figure S2 (g) for visualization. By having a smaller value in $\theta_7$, pixels that were originally over-saturated obtain normal saturation.

The last effect filter $f_8$ is the color temperature filter that shifts the image toward warmer or cooler colors according to the learnable parameter $\theta_8 \in \mathbb{R}^3$. This filter performs the computation of

$$\mathbf{I^{out}} = f_8(\mathbf{I^{in}}, \boldsymbol{\theta}_8) = \mathrm{Clamp}(\mathbf{I^{in}} + tile(\boldsymbol{\theta}_8)), \tag{20}$$

where $tile$ is the function that repeats the three dimensional parameter $\boldsymbol{\theta}_8$ into a shape identical to $\mathbf{I^{in}}$. See Figure S2 (h) for visualization. By having a larger value in the R channel of $\boldsymbol{\theta}_8$, the image would have a warmer color temperature, by having a larger value in the B channel of $\boldsymbol{\theta}_8$, the image would have a cooler temperature.

## D  Loss Function for TSO

In the second stage of TSO, we propose to use the loss function of

$$\mathcal{L}_{\mathrm{adv}}^{\mathrm{stage2}} = \mathcal{L}_{\mathrm{adv}} + \sum_{b=1}^{B} \sigma_b ||V(\mathrm{b}(\mathbf{P_{s1}}, \delta_{\mathbf{b}})) - V(\mathrm{b}(\mathbf{P_{s2}}, \delta_{\mathbf{b}}))||_2. \tag{21}$$

Within Equation (21), $\mathbf{P_{s1}}$ is the patch obtained in stage 1 (fixed in stage 2), $\mathbf{P_{s2}}$ is the patch to be optimized in stage 2 (initialized as $\mathbf{P_{s1}}$). The first term in Equation (21) is the adversarial attack loss

in equation 8, the b function is the Gaussian blur operation that extracts low frequency information, where the band width (or the range of low frequency information) extracted depends on the standard deviation $\delta_b$, $V$ is the function that vectorizes images, $\sigma_b$ is the parameter that controls the importance of maintaining patterns in different low frequency bands to be unchanged in the second stage. $B$ is the number of different low frequency bands to maintain within the second stage. By setting $\sigma_b$ corresponding to larger $\delta_b$ to larger values, the goal of maintaining the low frequency part within $\mathbf{P_{s1}}$ to be relatively unchanged in the second stage can be achieved.

# E   Analysis on Experiment Settings

To determine an appropriate experiment distance range, we assessed the detection success rates of YOLOv5 [21], Mask RCNN [14] and Deformable-DETR [43] across distances ranging from 3.5 to 50 meters. To demonstrate that the random patches can not influence the performance of the models, we include detection results on both normal clothing and results on subjects holding a random patch. The results are included in Figure S3, the patch used is illustrated in Figure 8 (b).

**Distance Range.** From Figure S3 (a), it can be observed that the performance of all models starts to slightly degrade at around 32 to 41 meters. Beyond 41 meters, the detection success rate of most models fell below 60%, failing to make robust decisions. To demonstrate our FDA method's effectiveness in performing attack, we conducted experiments within the 41 meters range.

**Influence of Holding a Patch.** Figure S3 shows that the performance of different detectors on normal clothing and random patches was comparable. The average detection success rate difference between the two cases is less than 7% across all models, where the difference typically occurs at distances beyond 41 meters where model performance is unstable. This result confirms that without an adversarial pattern, the patches could not reduce the detection success rates of different detectors within our selected distance range.

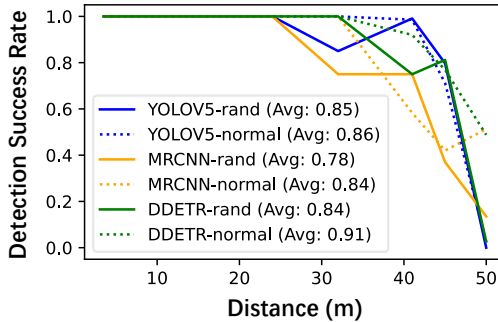

Figure S3: Evaluating the detection success rate of YOLOv5, Mask RCNN and Deformable-DETR on pedestrian subjects with normal clothing or holding random patches at different distances. "Normal" indicates experiments with normal clothing, "rand" indicates experiment with the pedestrian subject holding a random patch.

# F   Details for Datasets

**Distant Image Dataset.** When selecting the digital training and testing images of the distant image dataset, we selected images from different distributions. The training images were collected by cropping randomly generated color plates at different scales and different orientations. The testing images were obtained by extracting crops of randomly selected adversarial patterns targeted on the RetinaNet [37]. See Figure S4 (a) and (b) for training and testing images in the dataset. To prevent the DIC from overfitting to a certain day, we collected our distant image dataset under different skylights in 5 days. See Figure S4 (c) for different skylight samples. To calculate the skylight RGB value $\mathbf{P}^{\text{sky}}$ of a day, we took the image crop of the sky (near the horizon) and took an average over all pixels in the crop.

**Dataset for Pedestrian Attack.** The pedestrian images obtained from the COCO dataset, the INRIA pedestrian dataset and the PennFudan dataset are illustrated in Figure S4 (d), (e) and (f) respectively. The images collected contain pedestrians with different genders, races and ages. Samples of pedestrian images collected from online sources for increasing dataset diversity are demonstrated in Figure S4 (g).

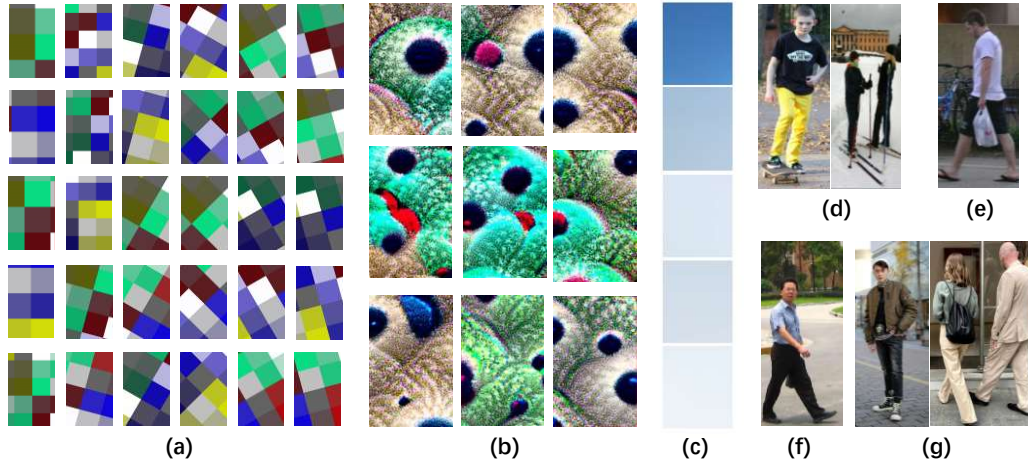

Figure S4: Samples from different datasets. (a) Training images of the distant image dataset. (b) Testing image of the distant image dataset. (c) The skylight samples collected when collecting the distant image dataset. (d) Samples of pedestrian images selected from the COCO dataset [24]. (e) Sample of pedestrian images selected from the INRIA dataset [3]. (f) Sample of pedestrian images selected from the Penn-Fudan dataset [36]. (g) Samples of pedestrian images extracted from online sources.

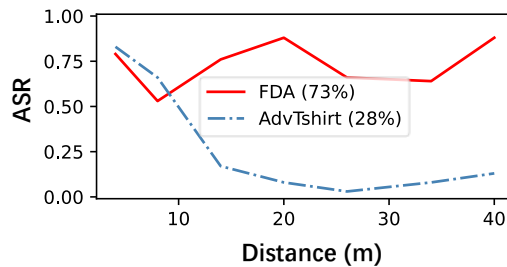

Figure S5: Digital world attack performance of FDA patch and AdvTshirt patch.

## G    FCN for Distant Image Conversion

For experiments in Section 5.1, we trained 7 FCNs, each dedicated to performing distant image conversion at specific testing distances. The choice of FCN architecture was motivated by its translation invariant property, which simulates the spatial invariant mapping performed by the physical world imaging pipeline. All FCNs used a three-layer design. The first and second layers were ReLU layers, each with 250 neurons, the output layers were Sigmoid layers with an output dimension of 3. These three output dimensions correspond to the RGB channels of the output images. The skylight RGB value $P^{sky}$, turbidity value $T$ were also fed into the kernel of the first layer. For FCNs dedicated to longer distances, we used kernels with larger width $w$ and stride $s$ in the first layer, allowing smaller output images to be encoded, simulating the effect of scale decrease as distance increases. More specifically, we set the $(w, s)$ pairs for the input layers to be (4,1), (7,3), (9,5), (13,7), (15,11), (18,13) and (21,15) for the distances of 4, 8, 14, 20, 26, 34 and 40 meters, respectively. To train the FCNs, we leveraged the training pipeline illustrated in Figure 4 (a).

## H    Digital World Results for FDA Patch Attack

To further demonstrate the effectiveness of our DIC simulation, we include the digital world attack result of the FDA patch and AdvTshirt patch in Figure S5. The two patches obtained average ASRs of 73% and 28% respectively, comparable to the corresponding physical world attack results in Figure 8.

| Patch | 5m (%) | 9m (%) | 15m (%) | 21m (%) | 28m (%) | 38m (%) | 45m (%) | 50m (%) | Avg. (%) |
|---|---|---|---|---|---|---|---|---|---|
| FDA | 39 | 41 | 98 | 93 | 97 | 73 | 65 | 100 | 76 |
| Rand | 0 | 0 | 0 | 0 | 15 | 1 | 20 | 100 | 17 |
| None | 0 | 0 | 0 | 0 | 0 | 2 | 29 | 98 | 16 |

Table S1: ASRs of different patches at distances neighboring the ones used in the main paper. "Rand" indicates tests performed with a random patch, "None" indicates no patch applied during the test.

## I Generalizing Across Different Distances

As demonstrated in Table S1, when we generalized the FDA pattern to distances different from the ones we used in the main paper, the FDA pattern obtained an average ASR of 76%, comparable to the average ASRs at the original testing distances. This result indicates that while our method optimizes the FDA pattern for equidistant points within the attack range, the design still allows the pattern to be effective across the entire range.

## J Ablation Study on FDA

**Ablation Study on DIC.** To evaluate the contribution of different design component within the DIC, following the testing configurations in Section 5.1, we evaluated the average $l_2$-norm error of the DIC across different distances with different design components removed one after another. With the full DIC, the average $l_2$-norm error was 0.11. With the effect filter simulation module, the camera simulation module and the atmospheric perspective module removed, the average $l_2$-norm error of the DIC increased to 0.12, 0.13 and 0.14 respectively. The result confirms that all design components within the DIC have their contribution to performing better distant image conversion.

**Ablation Study on MFO.** To evaluate the contribution of different design components within the MFO, we obtained the average ASRs of FDA patterns optimized with different MFO components removed using digital world tests. Empirically, we found that with the complete MFO design, the FDA pattern optimized obtained an average ASR of 73%. With the MSC removed, the average ASR of the resulting FDA pattern decreased to 66%. With the TSO further removed, the average ASR of the resulting FDA pattern decreased to 56%. The results confirm that all design components within the MFO have their contribution to boosting FDA pattern performance.

**Presence of Conflict in Attack Performance.** To demonstrate there is a conflict in attack performance between short and long distances that is impeding FDA pattern optimization, we optimized two FDA patterns without MFO (with identical optimization configuration but different initialization) and compared their performance against the pattern optimized with MFO using digital world tests. From Figure S6, it can be observed that without MFO, the two patches with different initialization performed better either at long distances or short distances and obtained relatively low average ASRs of 56% and 43%, demonstrates that the conflict in performance between short and

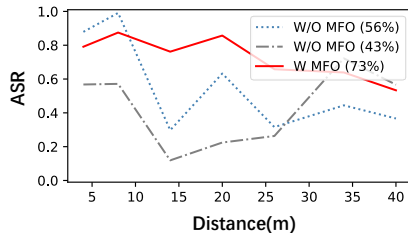

Figure S6: Digital world performance of patterns obtained with or without MFO.

long distances exists. This result demonstrates that the MFO successfully boosted FDA performance by resolving the conflict.

## K More Details on Adversarial Clothing

**Adversarial Clothing Tailoring.** Figure S7 (a) illustrates the adversarial clothing tailoring pipeline in the physical world. That is, the adversarial patches were first tiled and scaled so that their scale relative to the pedestrian subjects match the corresponding scale used when optimizing the patterns. Following the boxes indicated in Figure S7 (a), the crops for forming the trunk of the shirt, the sleeves of the shirt and the trousers were extracted. These crops were then printed onto clothes and tailored by professional tailors to create the final clothing. The adversarial patches optimized for FDA

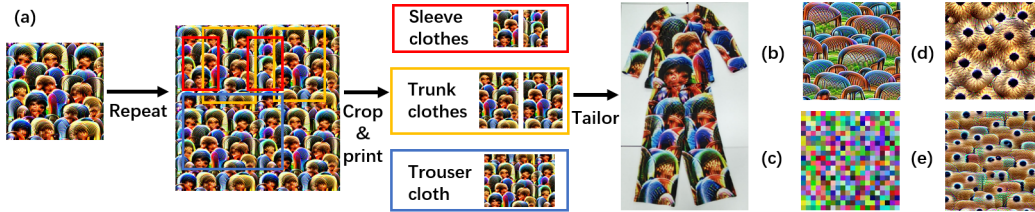

Figure S7: Adversarial clothing. (a) Clothing tailoring pipeline and FDA clothing pattern targeting YOLOV5. (b) TCA clothing adversarial pattern targeting YOLOV5. (c) Random clothing pattern. (d) FDA clothing pattern targeting Deformable DETR. (e) FDA clothing pattern targeting RetinaNet with PVT backbone.

| Target Model | IoU=0.5 (%) | IoU=0.4 (%) | IoU=0.3 (%) | IoU=0.2 (%) | IoU=0.1 (%) | IoU=0.0 (%) | Mean (%) |
|---|---|---|---|---|---|---|---|
| YOLOV5 [21] | 76 | 73 | 65 | 40 | 39 | 38 | 55 |
| D-DETR [43] | 71 | 70 | 68 | 68 | 68 | 68 | 69 |
| RetinaNet(PVT)[37] | 73 | 73 | 72 | 70 | 69 | 67 | 71 |

Table S2: Average ASRs (across different distances) of FDA clothing targeting different white-box models evaluated at different IoU thresholds. Mean indicates mean over Average ASRs at different IOU thresholds.

clothing targeting different detectors, TCA clothing targeting YOLOV5 and the random clothing are illustrated in Figure S7.

**FDA Clothing Performance Evaluated at Different IoU Thresholds.** In the main paper, we followed the convention from the baseline TCA paper [17] by using an IOU threshold of 0.5 to evaluate the performance of our FDA clothing. For readers interested in further details, we also include the average ASRs of the three clothing evaluated in the main paper at IOU thresholds of 0.5, 0.4, 0.3, 0.2, 0.1, and 0.0, with their corresponding mean values across different thresholds in Table S2. We found the average ASR of the FDA clothing to be stable across different IOU thresholds on most models. As the IOU threshold decreased, the average ASR of the FDA clothing targeting Deformable DETR and RetinaNet remained stable around 70%. However, when the target model was YOLOV5, the average ASR of the FDA clothing dropped from 76% to 38%. We found this drop to be due to the presence of small, abstract human-like patterns on the adversarial pattern (illustrated in Figure S7(a)). At small IOU thresholds, if the abstract human-like patterns were detected, the corresponding small bounding box for the pattern would cause the attack to be considered a failure, even when the actual pedestrian subjects were not detected. In contrast, the drop in performance did not occur for Deformable DETR and RetinaNet, since, as illustrated in Figure S7(d) and (e), the FDA patterns were formed by abstract Teddy bear nose patterns and abstract donut patterns, respectively.

